# Measuring Invariances in Deep Networks

**Ian J. Goodfellow, Quoc V. Le, Andrew M. Saxe, Honglak Lee, Andrew Y. Ng**
Computer Science Department
Stanford University
Stanford, CA 94305
{ia3n,quocle,asaxe,hllee,ang}@cs.stanford.edu

## Abstract

For many pattern recognition tasks, the ideal input feature would be invariant to multiple confounding properties (such as illumination and viewing angle, in computer vision applications). Recently, deep architectures trained in an unsupervised manner have been proposed as an automatic method for extracting useful features. However, it is difficult to evaluate the learned features by any means other than using them in a classifier. In this paper, we propose a number of empirical tests that directly measure the degree to which these learned features are invariant to different input transformations. We find that stacked autoencoders learn modestly increasingly invariant features with depth when trained on natural images. We find that convolutional deep belief networks learn substantially more invariant features in each layer. These results further justify the use of "deep" vs. "shallower" representations, but suggest that mechanisms beyond merely stacking one autoencoder on top of another may be important for achieving invariance. Our evaluation metrics can also be used to evaluate future work in deep learning, and thus help the development of future algorithms.

## 1   Introduction

Invariance to abstract input variables is a highly desirable property of features for many detection and classification tasks, such as object recognition. The concept of invariance implies a selectivity for complex, high level features of the input and yet a robustness to irrelevant input transformations. This tension between selectivity and robustness makes learning invariant features nontrivial. In the case of object recognition, an invariant feature should respond only to one stimulus despite changes in translation, rotation, complex illumination, scale, perspective, and other properties. In this paper, we propose to use a suite of "invariance tests" that directly measure the invariance properties of features; this gives us a measure of the quality of features learned in an unsupervised manner by a deep learning algorithm.

Our work also seeks to address the question: why are deep learning algorithms useful? Bengio and LeCun gave a theoretical answer to this question, in which they showed that a deep architecture is necessary to represent many functions compactly [1]. A second answer can also be found in such work as [2, 3, 4, 5], which shows that such architectures lead to useful representations for classification. In this paper, we give another, empirical, answer to this question: namely, we show that with increasing depth, the representations learned can also enjoy an increased degree of invariance. Our observations lend credence to the common view of invariances to minor shifts, rotations and deformations being learned in the lower layers, and being combined in the higher layers to form progressively more invariant features.

In computer vision, one can view object recognition performance as a measure of the invariance of the underlying features. While such an end-to-end system performance measure has many benefits, it can also be expensive to compute and does not give much insight into how to directly improve representations in each layer of deep architectures. Moreover, it cannot identify specific invariances

that a feature may possess. The test suite presented in this paper provides an alternative that can identify the robustness of deep architectures to specific types of variations. For example, using videos of natural scenes, our invariance tests measure the degree to which the learned representations are invariant to 2-D (in-plane) rotations, 3-D (out-of-plane) rotations, and translations. Additionally, such video tests have the potential to examine changes in other variables such as illumination. We demonstrate that using videos gives similar results to the more traditional method of measuring responses to sinusoidal gratings; however, the natural video approach enables us to test invariance to a wide range of transformations while the grating test only allows changes in stimulus position, orientation, and frequency.

Our proposed invariance measure is broadly applicable to evaluating many deep learning algorithms for many tasks, but the present paper will focus on two different algorithms applied to computer vision. First, we examine the invariances of stacked autoencoder networks [2]. These networks were shown by Larochelle et al. [3] to learn useful features for a range of vision tasks; this suggests that their learned features are significantly invariant to the transformations present in those tasks. Unlike the artificial data used in [3], however, our work uses natural images and natural video sequences, and examines more complex variations such as out-of-plane changes in viewing angle. We find that when trained under these conditions, stacked autoencoders learn increasingly invariant features with depth, but the effect of depth is small compared to other factors such as regularization. Next, we show that convolutional deep belief networks (CDBNs) [5], which are hand-designed to be invariant to certain local image translations, do enjoy dramatically increasing invariance with depth. This suggests that there is a benefit to using deep architectures, but that mechanisms besides simple stacking of autoencoders are important for gaining increasing invariance.

## 2    Related work

Deep architectures have shown significant promise as a technique for automatically learning features for recognition systems. Deep architectures consist of multiple layers of simple computational elements. By combining the output of lower layers in higher layers, deep networks can represent progressively more complex features of the input. Hinton et al. introduced the deep belief network, in which each layer consists of a restricted Boltzmann machine [4]. Bengio et al. built a deep network using an autoencoder neural network in each layer [2, 3, 6]. Ranzato et al. and Lee et al. explored the use of sparsity regularization in autoencoding energy-based models [7, 8] and sparse convolutional DBNs with probabilistic max-pooling [5] respectively. These networks, when trained subsequently in a discriminative fashion, have achieved excellent performance on handwritten digit recognition tasks. Further, Lee et al. and Raina et al. show that deep networks are able to learn good features for classification tasks even when trained on data that does not include examples of the classes to be recognized [5, 9].

Some work in deep architectures draws inspiration from the biology of sensory systems. The human visual system follows a similar hierarchical structure, with higher levels representing more complex features [10]. Lee et al., for example, compared the response properties of the second layer of a sparse deep belief network to V2, the second stage of the visual hierarchy [11]. One important property of the visual system is a progressive increase in the invariance of neural responses in higher layers. For example, in V1, complex cells are invariant to small translations of their inputs. Higher in the hierarchy in the medial temporal lobe, Quiroga et al. have identified neurons that respond with high selectivity to, for instance, images of the actress Halle Berry [12]. These neurons are remarkably invariant to transformations of the image, responding equally well to images from different perspectives, at different scales, and even responding to the text "Halle Berry." While we do not know exactly the class of all stimuli such neurons respond to (if tested on a larger set of images, they may well turn out to respond also to other stimuli than Halle Berry related ones), they nonetheless show impressive selectivity and robustness to input transformations.

Computational models such as the neocognitron [13], HMAX model [14], and Convolutional Network [15] achieve invariance by alternating layers of feature detectors with local pooling and subsampling of the feature maps. This approach has been used to endow deep networks with some degree of translation invariance [8, 5]. However, it is not clear how to explicitly imbue models with more complicated invariances using this fixed architecture. Additionally, while deep architectures provide a task-independent method of learning features, convolutional and max-pooling techniques are somewhat specialized to visual and audio processing.

# 3 Network architecture and optimization

We train all of our networks on natural images collected separately (and in geographically different areas) from the videos used in the invariance tests. Specifically, the training set comprises a set of still images taken in outdoor environments free from artificial objects, and was not designed to relate in any way to the invariance tests.

## 3.1 Stacked autoencoder

The majority of our tests focus on the stacked autoencoder of Bengio et al. [2], which is a deep network consisting of an autoencoding neural network in each layer. In the single-layer case, in response to an input pattern $x \in \mathbb{R}^n$, the activation of each neuron, $h_i$, $i = 1, \cdots, m$ is computed as

$$h(x) = \tanh\left(W_1 x + b_1\right),$$

where $h(x) \in \mathbb{R}^m$ is the vector of neuron activations, $W_1 \in \mathbb{R}^{m \times n}$ is a weight matrix, $b_1 \in \mathbb{R}^m$ is a bias vector, and tanh is the hyperbolic tangent applied componentwise. The network output is then computed as

$$\hat{x} = \tanh\left(W_2 h(x) + b_2\right),$$

where $\hat{x} \in \mathbb{R}^n$ is a vector of output values, $W_2 \in \mathbb{R}^{n \times m}$ is a weight matrix, and $b_2 \in \mathbb{R}^n$ is a bias vector. Given a set of $p$ input patterns $x^{(i)}$, $i = 1, \cdots, p$, the weight matrices $W_1$ and $W_2$ are adapted using backpropagation [16, 17, 18] to minimize the reconstruction error $\sum_{i=1}^{p} \left\| x^{(i)} - \hat{x}^{(i)} \right\|^2$.

Following [2], we successively train up layers of the network in a greedy layerwise fashion. The first layer receives a $14 \times 14$ patch of an image as input. After it achieves acceptable levels of reconstruction error, a second layer is added, then a third, and so on.

In some of our experiments, we use the method of [11], and constrain the expected activation of the hidden units to be sparse. We never constrain $W_1 = W_2^T$, although we found this to approximately hold in practice.

## 3.2 Convolutional Deep Belief Network

We also test a CDBN [5] that was trained using two hidden layers. Each layer includes a collection of "convolution" units as well as a collection of "max-pooling" units. Each convolution unit has a receptive field size of 10x10 pixels, and each max-pooling unit implements a probabilistic max-like operation over four (i.e., 2x2) neighboring convolution units, giving each max-pooling unit an overall receptive field size of 11x11 pixels in the first layer and 31x31 pixels in the second layer. The model is regularized in a way that the average hidden unit activation is sparse. We also use a small amount of $L_2$ weight decay.

Because the convolution units share weights and because their outputs are combined in the max-pooling units, the CDBN is explicitly designed to be invariant to small amounts of image translation.

# 4 Invariance measure

An ideal feature for pattern recognition should be both robust and selective. We interpret the hidden units as feature detectors that should respond strongly when the feature they represent is present in the input, and otherwise respond weakly when it is absent. An invariant neuron, then, is one that maintains a high response to its feature despite certain transformations of its input. For example, a face selective neuron might respond strongly whenever a face is present in the image; if it is invariant, it might continue to respond strongly even as the image rotates.

Building on this intuition, we consider hidden unit responses above a certain threshold to be *firing*, that is, to indicate the presence of some feature in the input. We adjust this threshold to ensure that the neuron is selective, and not simply always active. In particular we choose a separate threshold for each hidden unit such that all units fire at the same rate when presented with random stimuli. After identifying an input that causes the neuron to fire, we can test the robustness of the unit by calculating its firing rate in response to a set of transformed versions of that input.

More formally, a hidden unit $i$ is said to fire when $s_i h_i(x) > t_i$, where $t_i$ is a threshold chosen by our test for that hidden unit and $s_i \in \{-1, 1\}$ gives the sign of that hidden unit's values. The sign term $s_i$ is necessary because, in general, hidden units are as likely to use low values as to use high values to indicate the presence of the feature that they detect. We therefore choose $s_i$ to maximize the invariance score. For hidden units that are regularized to be sparse, we assume that $s_i = 1$, since their mean activity has been regularized to be low. We define the indicator function

$f_i(x) = 1\{s_i h_i(x) > t_i\}$, i.e., it is equal to one if the neuron fires in response to input $x$, and zero otherwise.

A *transformation function* $\tau(x, \gamma)$ transforms a stimulus $x$ into a new, related stimulus, where the degree of transformation is parametrized by $\gamma \in \mathbb{R}$. (One could also imagine a more complex transformation parametrized by $\gamma \in \mathbb{R}^n$.) In order for a function $\tau$ to be useful with our invariance measure, $|\gamma|$ should relate to the semantic dissimilarity between $x$ and $\tau(x, \gamma)$. For example, $\gamma$ might be the number of degrees by which $x$ is rotated.

A *local trajectory* $T(x)$ is a set of stimuli that are semantically similar to some reference stimulus $x$, that is

$$T(x) = \{\tau(x, \gamma) \mid \gamma \in \Gamma\}$$

where $\Gamma$ is a set of transformation amounts of limited size, for example, all rotations of less than 15 degrees.

The *global firing rate* is the firing rate of a hidden unit when applied to stimuli drawn randomly from a distribution $P(x)$:

$$G(i) = \mathbb{E}[f_i(x)],$$

where $P(x)$ is a distribution over the possible inputs $x$ defined for each implementation of the test.

Using these definitions, we can measure the robustness of a hidden unit as follows. We define the set $Z$ as a set of inputs that activate $h_i$ near maximally. The *local firing rate* is the firing rate of a hidden unit when it is applied to local trajectories surrounding inputs $z \in Z$ that maximally activate the hidden unit,

$$L(i) = \frac{1}{|Z|} \sum_{z \in Z} \frac{1}{|T(z)|} \sum_{x \in T(z)} f_i(x),$$

i.e., $L(i)$ is the proportion of transformed inputs that the neuron fires in response to, and hence is a measure of the robustness of the neuron's response to the transformation $\tau$.

Our invariance score for a hidden unit $h_i$ is given by

$$S(i) = \frac{L(i)}{G(i)}.$$

The numerator is a measure of the hidden unit's robustness to transformation $\tau$ near the unit's optimal inputs, and the denominator ensures that the neuron is selective and not simply always active.

In our tests, we tried to select the threshold $t_i$ for each hidden unit so that it fires one percent of the time in response to random inputs, that is, $G(i) = 0.01$. For hidden units that frequently repeat the same activation value (up to machine precision), it is sometimes not possible to choose $t_i$ such that $G(i) = 0.01$ exactly. In such cases, we choose the smallest value of $t(i)$ such that $G(i) > 0.01$.

Each of the tests presented in the paper is implemented by providing a different definition of $P(x)$, $\tau(x, \gamma)$, and $\Gamma$.

$S(i)$ gives the invariance score for a single hidden unit. The invariance score $Inv_p(N)$ of a network $N$ is given by the mean of $S(i)$ over the top-scoring proportion $p$ of hidden units in the deepest layer of $N$. We discard the $(1 - p)$ worst hidden units because different subpopulations of units may be invariant to different transformations. Reporting the mean of all unit scores would strongly penalize networks that discover several hidden units that are invariant to transformation $\tau$ but do not devote more than proportion $p$ of their hidden units to such a task.

Finally, note that while we use this metric to measure invariances in the visual features learned by deep networks, it could be applied to virtually any kind of feature in virtually any application domain.

## 5    Grating test

Our first invariance test is based on the response of neurons to synthetic images. Following such authors as Berkes et al.[19], we systematically vary the parameters used to generate images of gratings. We use as input an image $I$ of a grating, with image pixel intensities given by

$$I(x, y) = b + a \sin\left(\omega(x \cos(\theta) + y \sin(\theta) - \phi)\right),$$

where $\omega$ is the spatial frequency, $\theta$ is the orientation of the grating, and $\phi$ is the phase. To implement our invariance measure, we define $P(x)$ as a distribution over grating images. We measure invariance to translation by defining $\tau(x, \gamma)$ to change $\phi$ by $\gamma$. We measure invariance to rotation by defining $\tau(x, \gamma)$ to change $\omega$ by $\gamma$.[1]

## 6   Natural video test

While the grating-based invariance test allows us to systematically vary the parameters used to generate the images, it shares the difficulty faced by a number of other methods for quantifying invariance that are based on synthetic (or nearly synthetic) data [19, 20, 21]: it is difficult to generate data that systematically varies a large variety of image parameters.

Our second suite of invariance tests uses natural video data. Using this method, we will measure the degree to which various learned features are invariant to a wide range of more complex image parameters. This will allow us to perform quantitative comparisons of representations at each layer of a deep network. We also verify that the results using this technique align closely with those obtained with the grating-based invariance tests.

### 6.1   Data collection

Our dataset consists of natural videos containing common image transformations such as translations, 2-D (in-plane) rotations, and 3-D (out-of-plane) rotations. In contrast to labeled datasets like the NORB dataset [21] where the viewpoint changes in large increments between successive images, our videos are taken at sixty frames per second, and thus are suitable for measuring more modest invariances, as would be expected in lower layers of a deep architecture. After collection, the images are reduced in size to 320 by 180 pixels and whitened by applying a band pass filter. Finally, we adjust the constrast of the whitened images with a scaling constant that varies smoothly over time and attempts to make each image use as much of the dynamic range of the image format as possible. Each video sequence contains at least one hundred frames. Some video sequences contain motion that is only represented well near the center of the image; for example, 3-D (out-of-plane) rotation about an object in the center of the field of view. In these cases we cropped the videos tightly in order to focus on the relevant transformation.

### 6.2   Invariance calculation

To implement our invariance measure using natural images, we define $P(x)$ as a uniform distribution over image patches contained in the test videos, and $\tau(x, \gamma)$ to be the image patch at the same image location as $x$ but occurring $\gamma$ video frames later in time. We define $\Gamma = \{-5, \ldots, 5\}$. To measure invariance to different types of transformation, we simply use videos that involve each type of transformation. This obviates the need to define a complex $\tau$ capable of synthetically performing operations such as 3-D rotation.

## 7   Results
### 7.1   Stacked autoencoders
#### 7.1.1   Relationship between grating test and natural video test
Sinusoidal gratings are already used as a common reference stimulus. To validate our approach of using natural videos, we show that videos involving translation give similar test results to the phase variation grating test. Fig. 1 plots the invariance score for each of 378 one layer autoencoders regularized with a range of sparsity and weight decay parameters (shown in Fig. 3). We were not able to find as close of a correspondence between the grating orientation test and natural videos involving 2-D (in-plane) rotation. Our 2-D rotations were captured by hand-rotating a video camera in natural environments, which introduces small amounts of other types of transformations. To verify that the problem is not that rotation when viewed far from the image center resembles translation, we compare the invariance test scores for translation and for rotation in Fig. 2. The lack of any clear

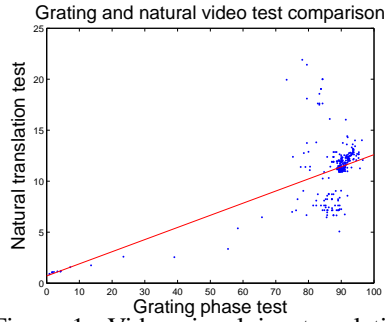

Figure 1: Videos involving translation give similar test results to synthetic videos of gratings with varying phase.

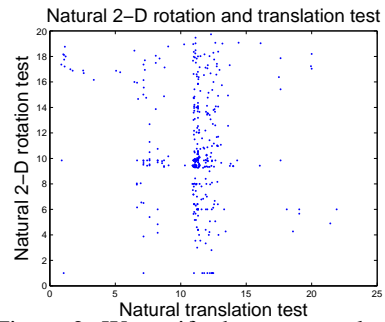

Figure 2: We verify that our translation and 2-D rotation videos do indeed capture different transformations.

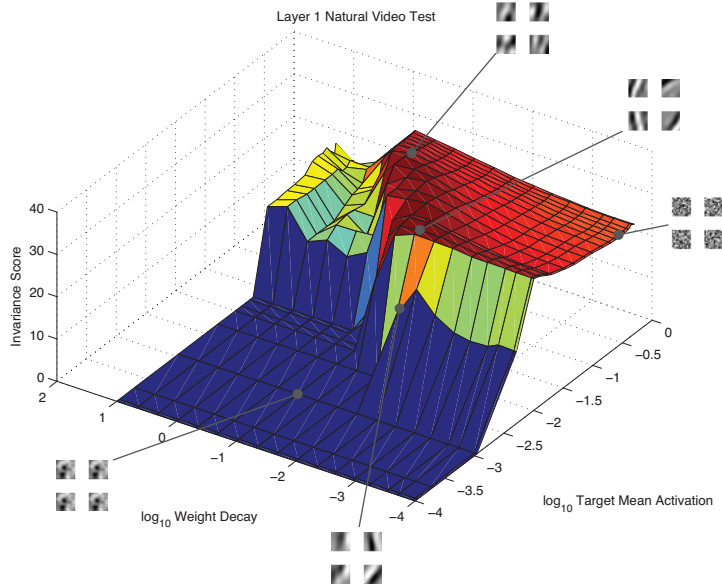

Figure 3: Our invariance measure selects networks that learn edge detectors resembling Gabor functions as the maximally invariant single-layer networks. Unregularized networks that learn high-frequency weights also receive high scores, but are not able to match the scores of good edge detectors. Degenerate networks in which every hidden unit learns essentially the same function tend to receive very low scores.

trend makes it obvious that while our 2-D rotation videos do not correspond exactly to rotation, they are certainly not well-approximated by translation.

### 7.1.2 Pronounced effect of sparsity and weight decay

We trained several single-layer autoencoders using sparsity regularization with various target mean activations and amounts of weight decay. For these experiments, we averaged the invariance scores of all the hidden units to form the network score, i.e., we used $p = 1$. Due to the presence of the sparsity regularization, we assume $s_i = 1$ for all hidden units. We found that sparsity and weight decay have a large effect on the invariance of a single-layer network. In particular, there is a semi-circular ridge trading sparsity and weight decay where invariance scores are high. We interpret this to be the region where the problem is constrained enough that the autoencoder must throw away some information, but is still able to extract meaningful patterns from its input. These results are visualized in Fig. 3. We find that a network with no regularization obtains a score of 25.88, and the best-scoring network receives a score of 32.41.

### 7.1.3 Modest improvements with depth

To investigate the effect of depth on invariance, we chose to extensively cross-validate several depths of autoencoders using only weight decay. The majority of successful image classification results in

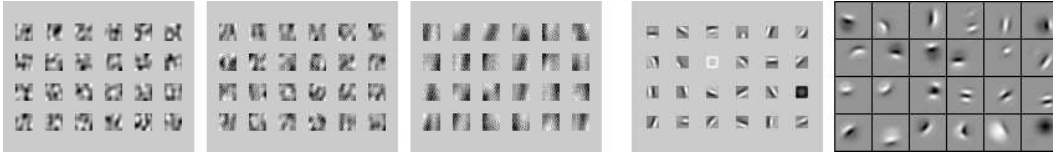

Figure 4: Left to right: weight visualizations from layer 1, layer 2, and layer 3 of the autoencoders; layer 1 and layer 2 of the CDBN. Autoencoder weight images are taken from the best autoencoder at each depth. All weight images are contrast normalized independently but plotted on the same spatial scale. Weight images in deeper layers are formed by making linear combinations of weight images in shallower layers. This approximates the function computed by each unit as a linear function.

the literature do not use sparsity, and cross-validating only a single parameter frees us to sample the search space more densely. We trained a total of $7^3$ networks with weight decay at each layer set to a value from $\{10, 1, 10^{-1}, 10^{-2}, 10^{-3}, 10^{-5}, 0\}$. For these experiments, we averaged the invariance scores of the top $20\%$ of the hidden units to form the network score, i.e., we used $p = .2$, and chose $s_i$ for each hidden unit to maximize the invariance score, since there was no sparsity regularization to impose a sign on the hidden unit values.

After performing this grid search, we trained 100 additional copies of the network with the best mean invariance score at each depth, holding the weight decay parameters constant and varying only the random weights used to initialize training. We found that the improvement with depth was highly significant statistically (see Fig. 5). However, the magnitude of the increase in invariance is limited compared to the increase that can be gained with the correct sparsity and weight decay.

## 7.2 Convolutional Deep Belief Networks

We also ran our invariance tests on a two layer CDBN. This provides a measure of the effectiveness of hard-wired techniques for achieving invariance, including convolution and max-pooling. The results are summarized in Table 1. These results cannot be compared directly to the results for autoencoders, because of the different receptive field sizes. The receptive field sizes in the CDBN are smaller than those in the autoencoder for the lower layers, but larger than those in the autoencoder for the higher layers due to the pooling effect. Note that the greatest relative improvement comes in the natural image tests, which presumably require greater sophistication than the grating tests. The single test with the greatest relative improvement is the 3-D (out-of-plane) rotation test. This is the most complex transformation included in our tests, and it is where depth provides the greatest percentagewise increase.

## 8 Discussion and conclusion

In this paper, we presented a set of tests for measuring invariances in deep networks. We defined a general formula for a test metric, and demonstrated how to implement it using synthetic grating images as well as natural videos which reveal more types of invariances than just 2-D (in-plane) rotation, translation and frequency.

At the level of a single hidden unit, our firing rate invariance measure requires learned features

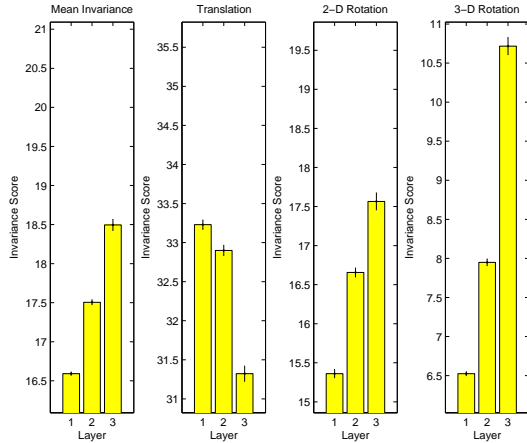

Figure 5: To verify that the improvement in invariance score of the best network at each layer is an effect of the network architecture rather than the random initialization of the weights, we retrained the best network of each depth 100 times. We find that the increase in the mean is statistically significant with $p < 10^{-60}$. Looking at the scores for individual invariances, we see that the deeper networks trade a small amount of translation invariance for a larger amount of 2-D (in-plane) rotation and 3-D (out-of-plane) rotation invariance. All plots are on the same scale but with different baselines so that the worst invariance score appears at the same height in each plot.

to balance high local firing rates with low global firing rates. This concept resembles the trade-off between precision and recall in a detection problem. As learning algorithms become more

| Test | Layer 1 | Layer 2 | % change |
|---|---|---|---|
| Grating phase | 68.7 | 95.3 | 38.2 |
| Grating orientation | 52.3 | 77.8 | 48.7 |
| Natural translation | 15.2 | 23.0 | 51.0 |
| Natural 3-D rotation | 10.7 | 19.3 | 79.5 |

Table 1: Results of the CDBN invariance tests.

advanced, another appropriate measure of invariance may be a hidden unit's invariance to object identity. As an initial step in this direction, we attempted to score hidden units by their mutual information with categories in the Caltech 101 dataset [22]. We found that none of our networks gave good results. We suspect that current learning algorithms are not yet sophisticated enough to learn, from only natural images, individual features that are highly selective for specific Caltech 101 categories, but this ability will become measurable in the future.

At the network level, our measure requires networks to have at least some subpopulation of hidden units that are invariant to each type of transformation. This is accomplished by using only the top-scoring proportion $p$ of hidden units when calculating the network score. Such a qualification is necessary to give high scores to networks that decompose the input into separate variables. For example, one very useful way of representing a stimulus would be to use some subset of hidden units to represent its orientation, another subset to represent its position, and another subset to represent its identity. Even though this would be an extremely powerful feature representation, a value of $p$ set too high would result in penalizing some of these subsets for not being invariant.

We also illustrated extensive findings made by applying the invariance test on computer vision tasks. However, the definition of our metric is sufficiently general that it could easily be used to test, for example, invariance of auditory features to rate of speech, or invariance of textual features to author identity.

A surprising finding in our experiments with visual data is that stacked autoencoders yield only modest improvements in invariance as depth increases. This suggests that while depth is valuable, mere stacking of shallow architectures may not be sufficient to exploit the full potential of deep architectures to learn invariant features.

Another interesting finding is that by incorporating sparsity, networks can become more invariant. This suggests that, in the future, a variety of mechanisms should be explored in order to learn better features. For example, one promising approach that we are currently investigating is the idea of learning slow features [19] from temporal data.

We also document that explicit approaches to achieving invariance such as max-pooling and weight-sharing in CDBNs are currently successful strategies for achieving invariance. This is not suprising given the fact that invariance is hard-wired into the network, but it validates the fact that our metric faithfully measures invariances. It is not obvious how to extend these explicit strategies to become invariant to more intricate transformations like large-angle out-of-plane rotations and complex illumination changes, and we expect that our metrics will be useful in guiding efforts to develop learning algorithms that automatically discover much more invariant features without relying on hard-wired strategies.

**Acknowledgments** This work was supported in part by the National Science Foundation under grant EFRI-0835878, and in part by the Office of Naval Research under MURI N000140710747. Andrew Saxe is supported by a Scott A. and Geraldine D. Macomber Stanford Graduate Fellowship. We would also like to thank the anonymous reviewers for their helpful comments.

## Footnotes

[1]Details: We define $P(x)$ as a uniform distribution over patches produced by varying $\omega \in \{2, 4, 6, 8\}$, $\theta \in \{0, \cdots, \pi\}$ in steps of $\pi/20$, and $\phi \in \{0, \cdots, \pi\}$ in steps of $\pi/20$. After identifying a grating that strongly activates the neuron, further local gratings $T(x)$ are generated by varying one parameter while holding all other optimal parameters fixed. For the translation test, local trajectories $T(x)$ are generated by modifying $\phi$ from the optimal value $\phi_{opt}$ to $\phi = \phi_{opt} \pm \{0, \cdots, \pi\}$ in steps of $\pi/20$, where $\phi_{opt}$ is the optimal grating phase shift. For the rotation test, local trajectories $T(x)$ are generated by modifying $\theta$ from the optimal value $\theta_{opt}$ to $\theta = \theta_{opt} \pm \{0, \cdots, \pi\}$ in steps of $\pi/40$, where $\theta_{opt}$ is the optimal grating orientation.

# References

[1] Y. Bengio and Y. LeCun. Scaling learning algorithms towards ai. In L. Bottou, O. Chapelle, D. DeCoste, and J. Weston, editors, *Large-Scale Kernel Machines*. MIT Press, 2007.

[2] Y. Bengio, P. Lamblin, D. Popovici, and H. Larochelle. Greedy layer-wise training of deep networks. In *NIPS*, 2007.

[3] H. Larochelle, D. Erhan, A. Courville, J. Bergstra, and Y. Bengio. An empirical evaluation of deep architectures on problems with many factors of variation. *ICML*, pages 473–480, 2007.

[4] G.E. Hinton, S. Osindero, and Y.-W. Teh. A fast learning algorithm for deep belief nets. *Neural Computation*, 18(7):1527–1554, 2006.

[5] H. Lee, R. Grosse, R. Ranganath, and A.Y. Ng. Convolutional deep belief networks for scalable unsupervised learning of hierarchical representations. In *ICML*, 2009.

[6] H. Larochelle, Y. Bengio, J. Louradour, and P. Lamblin. Exploring strategies for training deep neural networks. *The Journal of Machine Learning Research*, pages 1–40, 2009.

[7] M. Ranzato, Y-L. Boureau, and Y. LeCun. Sparse feature learning for deep belief networks. In *NIPS*, 2007.

[8] M. Ranzato, F.-J. Huang, Y-L. Boureau, and Y. LeCun. Unsupervised learning of invariant feature hierarchies with applications to object recognition. In *CVPR*. IEEE Press, 2007.

[9] Rajat Raina, Alexis Battle, Honglak Lee, Benjamin Packer, and Andrew Y. Ng. Self-taught learning: Transfer learning from unlabeled data. In *ICML '07: Proceedings of the 24th international conference on Machine learning*, 2007.

[10] D.J. Felleman and D.C. Van Essen. Distributed hierarchical processing in the primate cerebral cortex. *Cerebral Cortex*, 1(1):1–47, 1991.

[11] H. Lee, C. Ekanadham, and A.Y. Ng. Sparse deep belief network model for visual area v2. In *NIPS*, 2008.

[12] R. Quian Quiroga, L. Reddy, G. Kreiman, C. Koch, and I. Fried. Invariant visual representation by single neurons in the human brain. *Nature*, 435:1102–1107, 2005.

[13] K. Fukushima and S. Miyake. Neocognitron: A new algorithm for pattern recognition tolerant of deformations and shifts in position. *Pattern Recognition*, 1982.

[14] M. Riesenhuber and T. Poggio. Hierarchical models of object recognition in cortex. *Nature neuroscience*, 2(11):1019–1025, 1999.

[15] Y. LeCun, B. Boser, J.S. Denker, D. Henderson, R.E. Howard, W. Hubbard, and L.D. Jackel. Backpropagation applied to handwritten zip code recognition. *Neural Computation*, 1:541–551, 1989.

[16] P. Werbos. *Beyond regression: New tools for prediction and analysis in the behavioral sciences*. PhD thesis, Harvard University, 1974.

[17] Y. LeCun. Une procédure d'apprentissage pour réseau a seuil asymmetrique (a learning scheme for asymmetric threshold networks). In *Proceedings of Cognitiva 85*, pages 599–604, Paris, France, 1985.

[18] D.E. Rumelhart, G.E. Hinton, and R.J. Williams. Learning representations by back-propagating errors. *Nature*, 323:533–536, 1986.

[19] P. Berkes and L. Wiskott. Slow feature analysis yields a rich repertoire of complex cell properties. *Journal of Vision*, 5(6):579–602, 2005.

[20] L. Wiskott and T. Sejnowski. Slow feature analysis: Unsupervised learning of invariances. *Neural Computation*, 14(4):715–770, 2002.

[21] Y. LeCun, F.J. Huang, and L. Bottou. Learning methods for generic object recognition with invariance to pose and lighting. In *CVPR*, 2004.

[22] Li Fei-Fei, Rod Fergus, and Pietro Perona. Learning generative visual models from few training examples: An incremental bayesian approach tested on 101 object categories. page 178, 2004.

